# Link Prediction in Relational Data

**Ben Taskar    Ming-Fai Wong    Pieter Abbeel    Daphne Koller**
{*btaskar, mingfai.wong, abbeel, koller*}*@cs.stanford.edu*
Stanford University

## Abstract

Many real-world domains are *relational* in nature, consisting of a set of objects related to each other in complex ways. This paper focuses on predicting the existence and the type of links between entities in such domains. We apply the *relational Markov network* framework of Taskar *et al.* to define a joint probabilistic model over the entire link graph — entity attributes and links. The application of the RMN algorithm to this task requires the definition of probabilistic patterns over subgraph structures. We apply this method to two new relational datasets, one involving university webpages, and the other a social network. We show that the collective classification approach of RMNs, and the introduction of subgraph patterns over link labels, provide significant improvements in accuracy over flat classification, which attempts to predict each link in isolation.

## 1   Introduction

Many real world domains are richly structured, involving entities of multiple types that are related to each other through a network of different types of links. Such data poses new challenges to machine learning. One challenge arises from the task of predicting which entities are related to which others and what are the types of these relationships. For example, in a data set consisting of a set of hyperlinked university webpages, we might want to predict not just which page belongs to a professor and which to a student, but also which professor is which student's advisor. In some cases, the existence of a relationship will be predicted by the presence of a hyperlink between the pages, and we will have only to decide whether the link reflects an advisor-advisee relationship. In other cases, we might have to infer the very existence of a link from indirect evidence, such as a large number of co-authored papers. In a very different application, we might want to predict links representing participation of individuals in certain terrorist activities.

One possible approach to this task is to consider the presence and/or type of the link using only attributes of the potentially linked entities and of the link itself. For example, in our university example, we might try to predict and classify the link using the words on the two webpages, and the anchor words on the link (if present). This approach has the advantage that it reduces to a simple classification task and we can apply standard machine learning techniques. However, it completely ignores a rich source of information that is unique to this task — the graph structure of the link graph. For example, a strong predictor of an advisor-advisee link between a professor and a student is the fact that they jointly participate in several projects. In general, the link graph typically reflects common patterns of interactions between the entities in the domain. Taking these patterns into consideration should allow us to provide a much better prediction for links.

In this paper, we tackle this problem using the *relational Markov network (RMN)* framework of Taskar *et al.* [14]. We use this framework to define a single probabilistic model over the entire link graph, including both object labels (when relevant) and links between

objects. The model parameters are trained discriminatively, to maximize the probability of the (object and) link labels given the known attributes (e.g., the words on the page, hyperlinks). The learned model is then applied, using probabilistic inference, to predict and classify links using any observed attributes and links.

## 2   Link Prediction

A relational domain is described by a *relational schema*, which specifies a set of object types and attributes for them. In our web example, we have a Webpage type, where each page has a binary-valued attribute for each word in the dictionary, denoting whether the page contains the word. It also has an attribute representing the "class" of the webpage, e.g., a professor's homepage, a student's homepage, etc.

To address the link prediction problem, we need to make links first-class citizens in our model. Following [5], we introduce into our schema object types that correspond to links between entities. Each link object $\ell$ is associated with a tuple of entity objects $(o_1, \ldots, o_k)$ that participate in the link. For example, a Hyperlink link object would be associated with a pair of entities — the linking page, and the linked-to page, which are part of the link definition. We note that link objects may also have other attributes; e.g., a hyperlink object might have attributes for the anchor words on the link.

As our goal is to predict link existence, we must consider links that exist and links that do not. We therefore consider a set of *potential* links between entities. Each potential link is associated with a tuple of entity objects, but it may or may not actually exist. We denote this event using a binary *existence* attribute *Exists*, which is *true* if the link between the associated entities exists and *false* otherwise. In our example, our model may contain a potential link $\ell$ for each pair of webpages, and the value of the variable $\ell$.*Exists* determines whether the link actually exists or not. The link prediction task now reduces to the problem of predicting the existence attributes of these link objects.

An *instantiation* $\mathcal{I}$ specifies the set of entities of each entity type and the values of all attributes for all of the entities. For example, an instantiation of the hypertext schema is a collection of webpages, specifying their labels, the words they contain, and which links between them exist. A partial instantiation specifies the set of objects, and values for some of the attributes. In the link prediction task, we might observe all of the attributes for all of the objects, except for the existence attributes for the links. Our goal is to predict these latter attributes given the rest.

## 3   Relational Markov Networks

We begin with a brief review of the framework of undirected graphical models or *Markov Networks* [13], and their extension to relational domains presented in [14].

Let $\mathbf{V}$ denote a set of discrete random variables and $\mathbf{v}$ an assignment of values to $\mathbf{V}$. A Markov network for $\mathbf{V}$ defines a joint distribution over $\mathbf{V}$. It consists of an undirected dependency graph, and a set of parameters associated with the graph. For a graph $G$, a *clique* $c$ is a set of nodes $\mathbf{V}_c$ in $G$, not necessarily maximal, such that each $V_i, V_j \in \mathbf{V}_c$ are connected by an edge in $G$. Each clique $c$ is associated with a *clique potential* $\phi_c(\mathbf{V}_c)$, which is a non-negative function defined on the joint domain of $\mathbf{V}_c$. Letting $C(G)$ be the set of cliques, the Markov network defines the distribution $P(\mathbf{v}) = \frac{1}{Z} \prod_{c \in C(G)} \phi_c(\mathbf{v}_c)$, where $Z$ is the standard normalizing *partition function*.

A *relational Markov network (RMN)* [14] specifies the cliques and potentials between attributes of related entities at a template level, so a single model provides a coherent distribution for any collection of instances from the schema. RMNs specify the cliques using the notion of a *relational clique template*, which specify tuples of variables in the instantiation using a relational query language. (See [14] for details.)

For example, if we want to define cliques between the class labels of linked pages, we might define a clique template that applies to all pairs *page1*,*page2* and *link* of types

Webpage, Webpage and Hyperlink, respectively, such that *link* points from *page1* to *page2*. We then define a potential template that will be used for all pairs of variables *page1.Category* and *page2.Category* for such *page1* and *page2*.

Given a particular instantiation $\mathcal{I}$ of the schema, the RMN $\mathcal{M}$ produces an *unrolled* Markov network over the attributes of entities in $\mathcal{I}$, in the obvious way. The cliques in the unrolled network are determined by the clique templates $C$. We have one clique for each $c \in C(\mathcal{I})$, and all of these cliques are associated with the same clique potential $\phi_C$.

Taskar *et al.* show how the parameters of an RMN over a fixed set of clique templates can be learned from data. In this case, the training data is a single instantiation $\mathcal{I}$, where the same parameters are used multiple times — once for each different entity that uses a feature. A choice of clique potential parameters $\mathbf{w}$ specifies a particular RMN, which induces a probability distribution $P_{\mathbf{w}}$ over the unrolled Markov network.

Gradient descent over $\mathbf{w}$ is used to optimize the conditional likelihood of the target variables given the observed variables in the training set. The gradient involves a term which is the posterior probability of the target variables given the observed, whose computation requires that we run probabilistic inference over the entire unrolled Markov network. In relational domains, this network is typically large and densely connected, making exact inference intractable. Taskar *et al.* therefore propose the use of belief propagation [13, 17].

## 4  Subgraph Templates in a Link Graph

The structure of link graphs has been widely used to infer importance of documents in scientific publications [4] and hypertext (PageRank [12], Hubs and Authorities [8]). Social networks have been extensively analyzed in their own right in order to quantify trends in social interactions [16]. Link graph structure has also been used to improve document classification [7, 6, 15].

In our experiments, we found that the combination of a relational language with a probabilistic graphical model provides a very flexible framework for modeling complex patterns common in relational graphs. First, as observed by Getoor *et al.* [5], there are often correlations between the attributes of entities and the relations in which they participate. For example, in a social network, people with the same hobby are more likely to be friends.

We can also exploit correlations between the *labels* of entities and the relation type. For example, only students can be teaching assistants in a course. We can easily capture such correlations by introducing cliques that involve these attributes. Importantly, these cliques are informative even when attributes are not observed in the test data. For example, if we have evidence indicating an advisor-advisee relationship, our probability that X is a faculty member increases, and thereby our belief that X participates in a teaching assistant link with some entity Z decreases.

We also found it useful to consider richer subgraph templates over the link graph. One useful type of template is a *similarity* template, where objects that share a certain graph-based property are more likely to have the same label. Consider, for example, a professor X and two other entities Y and Z. If X's webpage mentions Y and Z in the same context, it is likely that the X-Y relation and the Y-Z relation are of the same type; for example, if Y is Professor X's advisee, then probably so is Z. Our framework accomodates these patterns easily, by introducing pairwise cliques between the appropriate relation variables.

Another useful type of subgraph template involves *transitivity* patterns, where the presence of an A-B link and of a B-C link increases (or decreases) the likelihood of an A-C link. For example, students often assist in courses taught by their advisor. Note that this type of interaction cannot be accounted for just using pairwise cliques. By introducing cliques over triples of relations, we can capture such patterns as well. We can incorporate even more complicated patterns, but of course we are limited by the ability of belief propagation to scale up as we introduce larger cliques and tighter loops in the Markov network.

We note that our ability to model these more complex graph patterns relies on our use

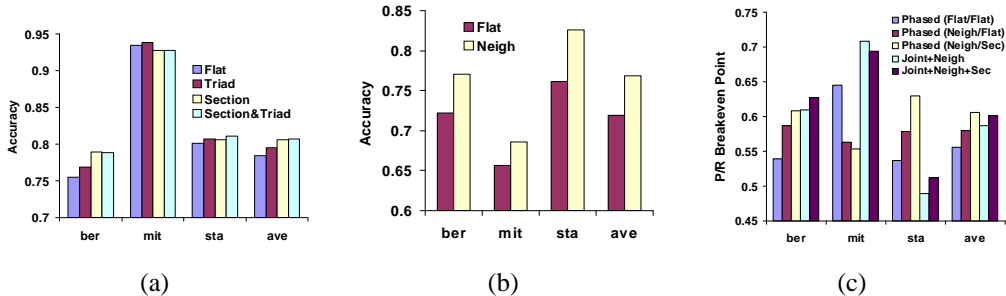

Figure 1: (a) Relation prediction with entity labels given. Relational models on average performed better than the baseline Flat model. (b) Entity label prediction. Relational model Neigh performed significantly better. (c) Relation prediction without entity labels. Relational models performed better most of the time, even though there are schools that some models performed worse.

of an undirected Markov network as our probabilistic model. In contrast, the approach of Getoor *et al.* uses directed graphical models (Bayesian networks and PRMs [9]) to represent a probabilistic model of both relations and attributes. Their approach easily captures the dependence of link existence on attributes of entities. But the constraint that the probabilistic dependency graph be a directed acyclic graph makes it hard to see how we would represent the subgraph patterns described above. For example, for the transitivity pattern, we might consider simply directing the correlation edges between link existence variables arbitrarily. However, it is not clear how we would then parameterize a link existence variable for a link that is involve in multiple triangles. See [15] for further discussion.

## 5 Experiments on Web Data

We collected and manually labeled a new relational dataset inspired by WebKB [2]. Our dataset consists of Computer Science department webpages from 3 schools: Stanford, Berkeley, and MIT. A total of $2954$ of pages are labeled into one of eight categories: faculty, student, research scientist, staff, research group, research project, course and organization (organization refers to any large entity that is not a research group). *Owned pages*, which are owned by an entity but are not the main page for that entity, were manually assigned to that entity. The average distribution of classes across schools is: organization (9%), student (40%), research group (8%), faculty (11%), course (16%), research project (7%), research scientist (5%), and staff (3%).

We established a set of candidate links between entities based on evidence of a relation between them. One type of evidence for a relation is a hyperlink from an entity page or one of its owned pages to the page of another entity. A second type of evidence is a *virtual link*: We assigned a number of aliases to each page using the page title, the anchor text of incoming links, and email addresses of the entity involved. Mentioning an alias of a page on another page constitutes a virtual link. The resulting set of 7161 candidate links were labeled as corresponding to one of five relation types — Advisor (faculty, student), Member (research group/project, student/faculty/research scientist), Teach (faculty/research scientist/staff, course), TA (student, course), Part-Of (research group, research proj) — or "none", denoting that the link does not correspond to any of these relations.

The observed attributes for each page are the words on the page itself and the "meta-words" on the page — the words in the title, section headings, anchors to the page from other pages. For links, the observed attributes are the anchor text, text just before the link (hyperlink or virtual link), and the heading of the section in which the link appears.

Our task is to predict the relation type, if any, for all the candidate links. We tried two settings for our experiments: with page categories observed (in the test data) and page categories unobserved. For all our experiments, we trained on two schools and tested on

the remaining school.

**Observed Entity Labels.** We first present results for the setting with observed page categories. Given the page labels, we can rule out many impossible relations; the resulting label breakdown among the candidate links is: none (38%), member (34%), part-of (4%), advisor (11%), teach (9%), TA (5%).

There is a huge range of possible models that one can apply to this task. We selected a set of models that we felt represented some range of patterns that manifested in the data.

Link-Flat is our baseline model, predicting links one at a time using multinomial logistic regression. This is a strong classifier, and its performance is competitive with other classifiers (e.g., support vector machines). The features used by this model are the labels of the two linked pages and the words on the links going from one page and its owned pages to the other page. The number of features is around $1000$.

The relational models try to improve upon the baseline model by modeling the interactions between relations and predicting relations jointly. The Section model introduces cliques over relations whose links appear consecutively in a section on a page. This model tries to capture the pattern that similarly related entities (e.g., advisees, members of projects) are often listed together on a webpage. This pattern is a type of similarity template, as described in Section 4. The Triad model is a type of transitivity template, as discussed in Section 4. Specifically, we introduce cliques over sets of three candidate links that form a triangle in the link graph. The Section + Triad model includes the cliques of the two models above.

As shown in Fig. 1(a), both the Section and Triad models outperform the flat model, and the combined model has an average accuracy gain of $2.26\%$, or $10.5\%$ relative reduction in error. As we only have three runs (one for each school), we cannot meaningfully analyze the statistical significance of this improvement.

As an example of the interesting inferences made by the models, we found a student-professor pair that was misclassified by the Flat model as none (there is only a single hyperlink from the student's page to the advisor's) but correctly identified by both the Section and Triad models. The Section model utilizes a paragraph on the student's webpage describing his research, with a section of links to his research groups and the link to his advisor. Examining the parameters of the Section model clique, we found that the model learned that it is likely for people to mention their research groups and advisors in the same section. By capturing this trend, the Section model is able to increase the confidence of the student-advisor relation. The Triad model corrects the same misclassification in a different way. Using the same example, the Triad model makes use of the information that both the student and the teacher belong to the same research group, and the student TAed a class taught by his advisor. It is important to note that none of the other relations are observed in the test data, but rather the model bootstraps its inferences.

**Unobserved Entity Labels.** When the labels of pages are not known during relations prediction, we cannot rule out possible relations for candidate links based on the labels of participating entities. Thus, we have many more candidate links that do not correspond to any of our relation types (e.g., links between an organization and a student). This makes the existence of relations a very low probability event, with the following breakdown among the potential relations: none (71%), member (16%), part-of (2%), advisor (5%), teach (4%), TA (2%). In addition, when we construct a Markov network in which page labels are not observed, the network is much larger and denser, making the (approximate) inference task much harder. Thus, in addition to models that try to predict page entity and relation labels simultaneously, we also tried a two-phase approach, where we first predict page categories, and then use the predicted labels as features for the model that predicts relations.

For predicting page categories, we compared two models. Entity-Flat model is multinomial logistic regression that uses words and "meta-words" from the page and its owned pages in separate "bags" of words. The number of features is roughly $10,000$. The Neighbors model is a relational model that exploits another type of similarity template: pages

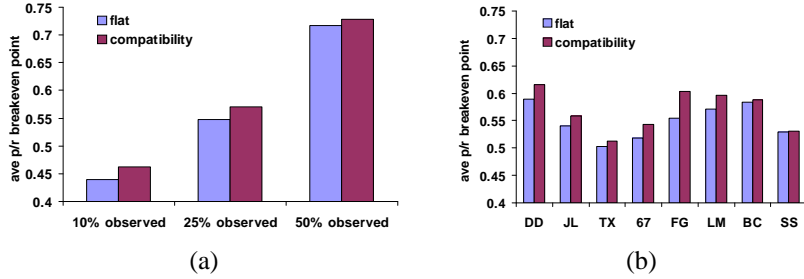

<div style="text-align:center">(a)  (b)</div>

Figure 2: (a) Average precision/recall breakeven point for 10%, 25%, 50% observed links. (b) Average precision/recall breakeven point for each fold of school residences at 25% observed links.

with similar urls often belong to the same category or tightly linked categories (research group/project, professor/course). For each page, two pages with urls closest in edit distance are selected as "neighbors", and we introduced pairwise cliques between "neighboring" pages. Fig. 1(b) shows that the Neighbors model clearly outperforms the Flat model across all schools, by an average of 4.9% accuracy gain.

Given the page categories, we can now apply the different models for link classification. Thus, the Phased (Flat/Flat) model uses the Entity-Flat model to classify the page labels, and then the Link-Flat model to classify the candidate links using the resulting entity labels. The Phased (Neighbors/Flat) model uses the Neighbors model to classify the entity labels, and then the Link-Flat model to classify the links. The Phased (Neighbors/Section) model uses the Neighbors to classify the entity labels and then the Section model to classify the links.

We also tried two models that predict page and relation labels simultaneously. The Joint + Neighbors model is simply the union of the Neighbors model for page categories and the Flat model for relation labels given the page categories. The Joint + Neighbors + Section model additionally introduces the cliques that appeared in the Section model between links that appear consecutively in a section on a page. We train the joint models to predict both page and relation labels simultaneously.

As the proportion of the "none" relation is so large, we use the probability of "none" to define a precision-recall curve. If this probability is less than some threshold, we predict the most likely label (other than none), otherwise we predict the most likely label (including none). As usual, we report results at the precision-recall breakeven point on the test data. Fig. 1(c) show the breakeven points achieved by the different models on the three schools. Relational models, both phased and joint, did better than flat models on the average. However, performance varies from school to school and for both joint and phased models, performance on one of the schools is worse than that of the flat model.

## 6 Experiments on Social Network Data

The second dataset we used has been collected by a portal website at a large university that hosts an online community for students [1]. Among other services, it allows students to enter information about themselves, create lists of their friends and browse the social network. Personal information includes residence, gender, major and year, as well as favorite sports, music, books, social activities, etc. We focused on the task of predicting the "friendship" links between students from their personal information and a subset of their links. We selected students living in sixteen different residences or dorms and restricted the data to the friendship links only within each residence, eliminating inter-residence links from the data to generate independent training/test splits. Each residence has about 15–25 students and an average student lists about 25% of his or her house-mates as friends.

We used an eight-fold train-test split, where we trained on fourteen residences and tested on two. Predicting links between two students from just personal information alone is a

very difficult task, so we tried a more realistic setting, where some proportion of the links is observed in the test data, and can be used as evidence for predicting the remaining links. We used the following proportions of observed links in the test data: 10%, 25%, and 50%. The observed links were selected at random, and the results we report are averaged over five folds of these random selection trials.

Using just the observed portion of links, we constructed the following flat features: for each student, the proportion of students in the residence that list him/her and the proportion of students he/she lists; for each pair of students, the proportion of other students they have as common friends. The values of the proportions were discretized into four bins. These features capture some of the relational structure and dependencies between links: Students who list (or are listed by) many friends in the observed portion of the links tend to have links in the unobserved portion as well. More importantly, having friends in common increases the likelihood of a link between a pair of students.

The Flat model uses logistic regression with the above features as well as personal information about each user. In addition to individual characteristics of the two people, we also introduced a feature for each match of a characteristic, for example, both people are computer science majors or both are freshmen.

The Compatibility model uses a type of similarity template, introducing cliques between each pair of links emanating from each person. Similarly to the Flat model, these cliques include a feature for each match of the characteristics of the two potential friends. This model captures the tendency of a person to have friends who share many characteristics (even though the person might not possess them). For example, a student may be friends with several CS majors, even though he is not a CS major himself. We also tried models that used transitivity templates, but the approximate inference with 3-cliques often failed to converge or produced erratic results.

Fig. 2(a) compares the average precision/recall breakpoint achieved by the different models at the three different settings of observed links. Fig. 2(b) shows the performance on each of the eight folds containing two residences each. Using a paired t-test, the Compatibility model outperforms Flat with p-values 0.0036, 0.00064 and 0.054 respectively.

## 7 Discussion and Conclusions

In this paper, we consider the problem of link prediction in relational domains. We focus on the task of collective link classification, where we are simultaneously trying to predict and classify an entire set of links in a link graph. We show that the use of a probabilistic model over link graphs allows us to represent and exploit interesting subgraph patterns in the link graph. Specifically, we have found two types of patterns that seem to be beneficial in several places. Similarity templates relate the classification of links or objects that share a certain graph-based property (e.g., links that share a common endpoint). Transitivity templates relate triples of objects and links organized in a triangle. We show that the use of these patterns significantly improve the classification accuracy over flat models.

Relational Markov networks are not the only method one might consider applying to the link prediction and classification task. We could, for example, build a link predictor that considers other links in the graph by converting graph features into flat features [11], as we did in the social network data. As our experiments show, even with these features, the collective prediction approach work better. Another approach is to use relational classifiers such as variants of *inductive logic programming* [10]. Generally, however, these methods have been applied to the problem of predicting or classifying a single link at a time. It is not clear how well they would extend to the task of simultaneously predicting an entire link graph. Finally, we could apply the directed PRM framework of [5]. However, as shown in [15], the discriminatively trained RMNs perform significantly better than generatively trained PRMs even on the simpler entity classification task. Furthermore, as we discussed, the PRM framework cannot represent (in any natural way) the type of subgraph patterns that seem prevalent in link graph data. Therefore, the RMN framework seems much more

appropriate for this task.

Although the RMN framework worked fairly well on this task, there is significant room for improvement. One of the key problems limiting the applicability of approach is the reliance on belief propagation, which often does not converge in more complex problems. This problem is especially acute in the link prediction problem, where the presence of all potential links leads to densely connected Markov networks with many short loops. This problem can be addressed with heuristics that focus the search on links that are plausible (as we did in a very simple way in the webpage experiments). A more interesting solution would be to develop a more integrated approximate inference / learning algorithm.

Our results use a set of relational patterns that we have discovered to be useful in the domains that we have considered. However, many other rich and interesting patterns are possible. Thus, in the relational setting, even more so than in simpler tasks, the issue of feature construction is critical. It is therefore important to explore the problem of automatic feature induction, as in [3].

Finally, we believe that the problem of modeling link graphs has numerous other applications, including: analyzing communities of people and hierarchical structure of organizations, identifying people or objects that play certain key roles, predicting current and future interactions, and more.

**Acknowledgments.** This work was supported by ONR Contract F3060-01-2-0564-P00002 under DARPA's EELD program. P. Abbeel was supported by a Siebel Grad. Fellowship.

# References

[1] L. Adamic, O. Buyukkokten, and E. Adar. A social network caught in the web. http://www.hpl.hp.com/shl/papers/social/, 2002.

[2] M. Craven, D. DiPasquo, D. Freitag, A. McCallum, T. Mitchell, K. Nigam, and S. Slattery. Learning to extract symbolic knowledge from the world wide web. In *Proc. AAAI*, 1998.

[3] S. Della Pietra, V. Della Pietra, and J. Lafferty. Inducing features of random fields. *IEEE Trans. on Pattern Analysis and Machine Intelligence*, 19(4):380–393, 1997.

[4] L. Egghe and R. Rousseau. *Introduction to Informetrics*. Elsevier, 1990.

[5] L. Getoor, N. Friedman, D. Koller, and B. Taskar. Probabilistic models of relational structure. In *Proc. ICML*, 2001.

[6] L. Getoor, E. Segal, B. Taskar, and D. Koller. Probabilistic models of text and link structure for hypertext classification. In *IJCAI Workshop on Text Learning: Beyond Supervision*, 2001.

[7] R. Ghani, S. Slattery, and Y. Yang. Hypertext categorization using hyperlink patterns and meta data. In *Proc ICML*, 2001.

[8] J. M. Kleinberg. Authoritative sources in a hyperlinked environment. *JACM*, 46(5):604–632, 1999.

[9] D. Koller and A. Pfeffer. Probabilistic frame-based systems. In *Proc. AAAI98*, pages 580–587, 1998.

[10] Nada Lavrač and Saso Džeroski. *Inductive Logic Programming: Techniques and Applications*. Ellis Horwood, 1994.

[11] J. Neville and D. Jensen. Iterative classification in relational data. In *AAAI Workshop on Learning Statistical Models from Relational Data*, 2000.

[12] L. Page, S. Brin, R. Motwani, and T. Winograd. The pagerank citation ranking: Bringing order to the web. Technical report, Stanford University, 1998.

[13] J. Pearl. *Probabilistic Reasoning in Intelligent Systems*. Morgan Kaufmann, 1988.

[14] B. Taskar, P. Abbeel, and D. Koller. Discriminative probabilistic models for relational data. In *Proc. UAI*, 2002.

[15] B. Taskar, E. Segal, and D. Koller. Probabilistic classification and clustering in relational data. In *Proc. IJCAI*, pages 870–876, 2001.

[16] S. Wasserman and P. Pattison. Logit models and logistic regression for social networks. *Psychometrika*, 61(3):401–425, 1996.

[17] J. Yedidia, W. Freeman, and Y. Weiss. Generalized belief propagation. In *Proc. NIPS*, 2000.
